# Extraction of temporal features in the electrosensory system of weakly electric fish

**Fabrizio Gabbiani***
Division of Biology
139-74 Caltech
Pasadena, CA 91125

**Walter Metzner**
Department of Biology
Univ. of Cal. Riverside
Riverside, CA 92521-0427

**Ralf Wessel**
Department of Biology
Univ. of Cal. San Diego
La Jolla, CA 92093-0357

**Christof Koch**
Division of Biology
139-74 Caltech
Pasadena, CA 91125

## Abstract

The encoding of random time-varying stimuli in single spike trains of electrosensory neurons in the weakly electric fish *Eigenmannia* was investigated using methods of statistical signal processing. At the first stage of the electrosensory system, spike trains were found to encode faithfully the detailed time-course of random stimuli, while at the second stage neurons responded specifically to features in the temporal waveform of the stimulus. Therefore stimulus information is processed at the second stage of the electrosensory system by extracting temporal features from the faithfully preserved image of the environment sampled at the first stage.

## 1 INTRODUCTION

The weakly electric fish, *Eigenmannia*, generates a quasi sinusoidal, dipole-like electric field at individually fixed frequencies ($250 - 600$ Hz) by discharging an electric organ located in its tail (see Bullock and Heilgenberg, 1986 for reviews). The fish sense local changes in the electric field by means of two types of tuberous electroreceptors located on the body surface. T-type electroreceptors fire phase-locked to the zero-crossing of the electric field once per cycle of the electric organ discharge

*email: gabbiani@klab.caltech.edu, wmetzner@mail.ucr.edu, rwessel@jeeves.ucsd.edu, koch@klab.caltech.edu.

(EOD) and are thus able to encode phase changes. P-type electroreceptors fire in a more loosely phase-locked manner with a probability smaller than 1 per EOD. Their probability of firing increases with the mean amplitude of the field thereby allowing them to encode amplitude changes (Zakon, 1986).

This information is used by the fish in order to locate objects (electrolocation, Bastian 1986) as well as for communication with conspecifics (Hopkins, 1988). One behavior which has been most thoroughly studied (Heiligenberg, 1991), the jamming avoidance response, occurs when two fish with similar EOD frequency (less than 15 Hz difference) approach close enough to sense each other's field. In order to minimize beat patterns resulting from their summed electric fields, the fish with the higher (resp. lower) EOD raises further (resp. lowers) its own EOD frequency. The resulting frequency difference increase reduces the distortions in the interfering EODs. The fish is known to correlate phase differences computed across body regions with local amplitude increases or decreases in order to determine whether it should raise or lower its own EOD.

At the level of the first central nervous nucleus of the electrosensory pathway, the electrosensory lateral line lobe of the hindbrain (ELL), phase and amplitude information are processed nearly independently of each other (Maler et al., 1981). Amplitude information is encoded in the spike trains of ELL pyramidal cells that receive input from P-receptor afferents and transmit their information to higher order levels of the electrosensory system. Two functional classes of pyramidal cells are distinguished: E-type pyramidal cells respond by raising their firing frequency to increases in the amplitude of an externally applied electric field while I-type pyramidal cells increase their firing frequency when the amplitude decreases (Bastian, 1981).

The aim of this study was to characterize the temporal information processing performed by ELL pyramidal cells on random electric field amplitude modulations and to relate it to the information carried by P-receptor afferents. To this end we recorded the responses of P-receptor afferents and pyramidal cells to random electric field amplitude modulations and analyzed them using two different methods: a signal estimation method characterizing to what extent the neuronal response encodes the detailed time-course of the stimulus and a signal-detection method developed to identify features encoded in spike trains. These two methods as well as the electrophysiology are explained in the next section followed by the result section and a short discussion.

## 2   METHODS

### 2.1   ELECTROPHYSIOLOGY

Adult specimens of *Eigenmannia* were immobilized by intramuscular injection of a curare-like drug (Flaxedil). This also strongly attenuated the fish's EODs. The fish were held in place by a foam-lined clamp in an experimental tank and an EOD substitute electric field was established by two electrodes placed in the mouth and near the tail of the fish. The carrier frequency of the electric field, $f_{carrier}$, was chosen equal to the EOD frequency prior to curarization and the voltage generating the stimulus was modulated according to

$$V(t) = A_0(1 + s(t))\cos(2\pi f_{carrier}),$$

where $A_0$ is the mean amplitude and $s(t)$ is a random, zero-mean modulation having a flat (white) spectrum up to a variable cut-off frequency $f_c$ and a variable standard deviation $\sigma$. The modulation $s(t)$ was generated by playing a blank tape on a tape

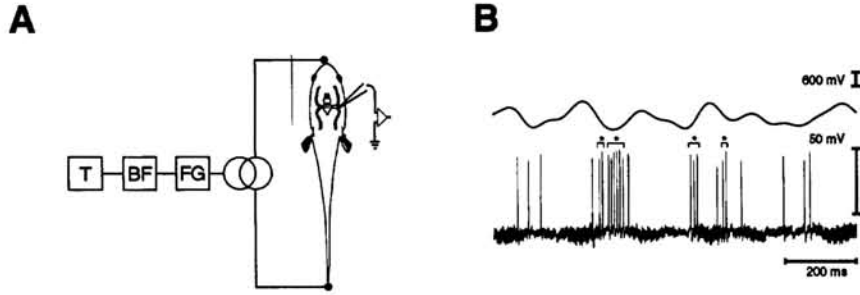

Figure 1: A. Schematic drawing of the experimental set-up. Tape recorder (T), variable cutoff frequency Bessel filter (BF) and function generator (FG). B. Sample amplitude modulation $s(t)$ and intracellular recording from a pyramidal cell (I-type, $f_c = 12$ Hz). Spikes occurring in bursts are marked with an asterisk (see sect. 2.3.2). The intracellular voltage trace reveals a high frequency noise caused by the EOD substitute signal and a high electrode resistance.

recorder, passing the signal through a variable cut-off frequency low-pass filter before multiplying it by the frequency carrier signal in a function generator (fig. 1A).

Extracellular recordings from P-receptor afferents were made by exposing the anterior lateral line nerve. Intracellular recordings from ELL pyramidal cells were obtained by positioning electrodes in the central region of the pyramidal cell layer. Intracellular recording electrodes were filled with neurotracer (neurobiotin) to be used for subsequent intracellular labeling if the recordings were stable long enough. This allowed to verify the cell type and its location within the ELL. In case no intracellular labeling could be made the recording site was verified by setting electrolytic lesions at the conclusion of the experiment. In the subsequent data analysis, data from E– and I–type pyramidal cells and from two different maps (centromedial and lateral, Carr et al., 1982) were pooled. For further experimental details, see Wessel et al. (1996), Metzner and Heiligenberg (1991), Metzner (1993).

## 2.2   SIGNAL ESTIMATION

The ability of single spike trains to carry detailed time-varying stimulus information was assessed by estimating the stimulus from the spike train. The spike trains, $x(t) = \sum \delta(t - t_i)$, where $t_i$ are the spike occurrence times, were convolved with a filter $h$ (Wiener-Kolmogorov filtering; Poor, 1994; Bialek et al. 1991),

$$s_{est}(t) = \int dt_1 \, h(t_1) x(t - t_1)$$

chosen in order to minimize the mean square error between the true stimulus and the estimated stimulus,

$$\epsilon^2 = \langle (s(t) - s_{est}(t))^2 \rangle.$$

The optimal filter $h(t)$ is determined in Fourier space as the ratio of the Fourier transform of the cross-correlation between stimulus and spike train, $R_{xs}(\tau) = \langle x(t)s(t+\tau) \rangle$ and the Fourier transform of the autocorrelation (power spectrum) of the spike train, $R_{xx}(\tau) = \langle x(t)x(t+\tau) \rangle$. The accuracy at which single spike trains transmit information about the stimulus was characterized in the time domain by the coding fraction, defined as $\gamma = 1 - \epsilon/\sigma$, were $\sigma$ is the standard deviation of the stimulus. The coding fraction is normalized between 1 when the stimulus is perfectly estimated by the spike train ($\epsilon = 0$) and 0, when the estimation performance of the spike train is at chance level ($\epsilon = \sigma$). Thus, the coding fraction can be

compared across experiments. For further details and references on this stimulus estimation method in the context of neuronal sensory processing, see Gabbiani and Koch (1996) and Gabbiani (1996).

## 2.3 FEATURE EXTRACTION

### 2.3.1 General procedure

The ability of single spikes to encode the presence of a temporal feature in the stimulus waveform was assessed by adapting a Fisher linear discriminant classification scheme to our data (Anderson, 1984; sect. 6.5). Each random stimulus wave-form and spike response of pyramidal cells (resp. P-receptor afferents) were binned. The bin size $\Delta$ was varied between $\Delta_{min} = 0.5$ ms, corresponding to the sampling ratio and $\Delta_{max}$, corresponding to the longest interval leading to a maximum of one spike per bin. The sampling interval yielding the best performance (see below) was finally retained. Typical bin sizes were $\Delta = 7$ ms for pyramidal cells (typical mean firing rate: 30 Hz) and $\Delta = 1$ ms for P-receptor afferents (typical firing rate: 200 Hz). The mean stimulus preceding a spike containing bin ($\mathbf{m_1}$) or no-spike containing bin ($\mathbf{m_0}$) as well as the covariances ($\mathbf{\Sigma_1}, \mathbf{\Sigma_0}$) of these distributions were computed (Anderson, 1984; sect. 3.2). Mean vectors (resp. covariance matrices) had at most 100 (resp. $100 \times 100$) components. The optimal linear feature vector $\mathbf{f}$ predicting the occurrence or non-occurrence of a spike was found by maximizing the signal-to-noise ratio (see fig. 2A and Poor, 1994; sect. IIIB)

$$\text{SNR}(\mathbf{f}) = \frac{[\mathbf{f} \cdot (\mathbf{m_1} - \mathbf{m_0})]^2}{\mathbf{f} \cdot \frac{1}{2}(\mathbf{\Sigma_0} + \mathbf{\Sigma_1})\mathbf{f}}. \tag{1}$$

The vector $\mathbf{f}$ is solution of $(\mathbf{m_1} - \mathbf{m_0}) = (\mathbf{\Sigma_0} + \mathbf{\Sigma_1})\mathbf{f}$. This equation was solved by diagonalizing $\mathbf{\Sigma_0} + \mathbf{\Sigma_1}$ and retaining the first n largest eigenvalues accounting for 99% of the variance (Jolliffe, 1986; sect. 6.1 and 8.1). The optimal feature vector $\mathbf{f}$ thus obtained corresponded to up– or downstrokes in the stimulus amplitude modulation for E– and I–type pyramidal cells respectively, as expected from their mean response properties to changes in the electric field amplitude (see sect. 1). Similarly, optimal feature vectors for P-receptor afferents corresponded to upstrokes in the electric field amplitude (see sect. 1).

Once $\mathbf{f}$ was determined, we projected the stimuli preceding a spike or no spike onto the optimal feature vector (fig. 2A) and computed the probability of correct classification between the two distributions so obtained by the resubstitution method (Raudys and Jain, 1991). The probability of correct classification ($P_{CC}$) is obtained by optimizing the value of the threshold used to separate the two distributions in order to maximize

$$P_{CC} = \frac{1}{2}(1 - P_{FA}) + \frac{1}{2}P_{CD}, \tag{2}$$

where the probabilities of false alarm ($P_{FA}$) and correct detection ($P_{CD}$) depend on the threshold.

### 2.3.2 Distinction between isolated spikes and burst-like spike patterns

A large fraction (56% ± 21%, n = 30) of spikes generated by pyramidal cells in response to random electric field amplitude modulations occurred in bursts (mean burst length: 18 ± 9 ms, mean number of spikes per burst: 2.9 ± 1.3, n = 30, fig. 1B). In order to verify whether spikes occurring in bursts corresponded to a more reliable encoding of the feature vector, we separated the distribution of stimuli occurring before a spike in two distributions, conditioned on whether the

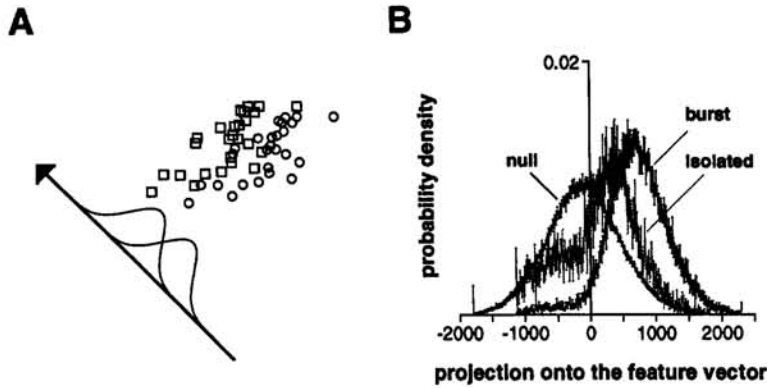

Figure 2: A. 2-dimensional example of two random distributions (circles and squares) as well as the optimal discrimination direction determined by maximizing the signal-to-noise ratio of eq. 1. The 1-dimensional projection of the two distributions onto the optimal direction is also shown (compare with B). B. Example of the distribution of stimuli projected onto the optimal feature vector (same cell as in fig. 1B). Stimuli preceding a bin containing no spike (null), an isolated spike (isolated) and a spike belonging to a burst (burst). Horizontal scale is arbitrary (see eq. 1).

spike belonged to a burst or not. The stimuli were then projected onto the feature vector (fig. 2B), as described in 2.3.1, and the probability of correct classification between the distribution of stimuli occurring before no spike and isolated spike bins was compared to the probability of correct classification between the distribution of stimuli occurring before no spike and burst spike bins (see sect. 3).

## 3   RESULTS

The results are summarized in fig. 3. Data were analyzed from 30 pyramidal cells (E– and I-type) and 20 P-receptor afferents for a range of stimulus parameters ($f_c = 2 - 40$ Hz, $\sigma = 0.1 - 0.4$, $A_0$ was varied in order to obtain $\pm20$ dB changes around the physiological value of the mean electric field amplitude which is of the order of 1 mV/cm). Fig. 3A reports the best probability of correct classification (eq. 2) obtained for each pyramidal cell (white squares) / P-receptor afferent (black dots) as a function of the coding fraction observed in the same experiment (note that for pyramidal cells only burst spikes are shown, see sect. 2.3.2 and fig. 3B). The horizontal axis shows that while the coding fraction of P-receptors afferents can be very high (up to 75% of the detailed stimulus time-course is encoded in a single spike train), pyramidal cells only poorly transmit information on the detailed time-course of the stimulus (less than 20% in most cases). In contrast, the vertical axis shows that pyramidal cells outperform P-receptor afferent in the classification task: it is possible to classify with up to 85% accuracy whether a given stimulus will cause a short burst of spikes or not by comparing it to a single feature vector **f**. This indicates that the presence of an up– or downstroke (the feature vector) is reliably encoded by pyramidal cells. Fig. 3B shows for each experiment on the ordinate the discrimination performance (eq. 2) for stimuli preceding isolated spikes against stimuli preceding no spike. The abscissa plots the discrimination performance (eq. 2) for stimuli preceding spikes occurring in bursts (white squares, fig. 3A) or stimuli preceding all spikes (black squares) against stimuli preceding

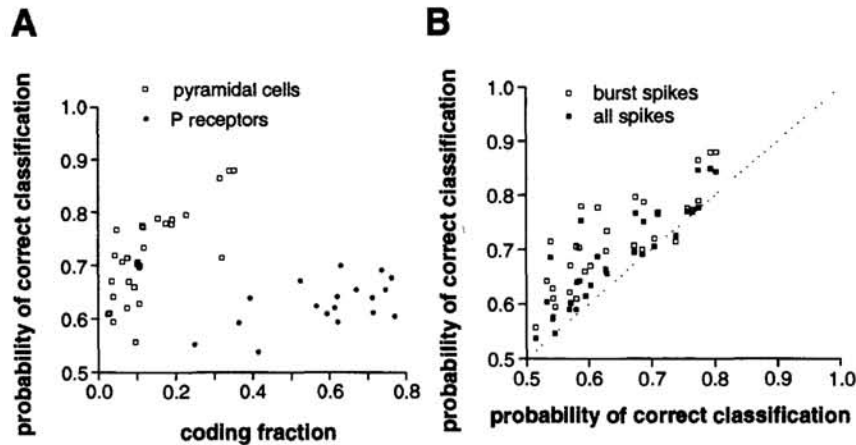

Figure 3: A. Coding fraction and probability of correct classification for pyramidal cells (white squares, burst spikes only) and P-receptor afferents (black circles). B. Probability of correct classification against stimuli preceding no spikes for stimuli preceding burst spikes or all spikes vs. stimuli preceding isolated spikes. Dashed line: identical performance.

no spike. The distribution of stimuli occurring before burst spikes (all spikes) is more easily distinguished from the distribution of stimuli occurring before no spike than the distribution of stimuli preceding isolated spikes. This clearly indicates that spikes occurring in bursts carry more reliable information than isolated spikes.

## 4  DISCUSSION

We have analyzed the response of P-receptor afferents and pyramidal cells to random electric field amplitude modulations using methods of statistical signal processing. The previously studied mean responses of P-receptor afferents and pyramidal cells to step amplitude changes or sinusoidal modulations of an externally applied electric field left several alternatives open for the encoding and processing of stimulus information in single spike trains. We find that, while P-receptor afferents encode reliably the detailed time-course of the stimulus, pyramidal cells do not. In contrast, pyramidal cells perform better than P-receptor afferents in discriminating the occurrence of up– and downstrokes in the amplitude modulation. The presence of these features is signaled most reliably to higher order stations in the electrosensory system by short bursts of spikes emitted by pyramidal cells in response to the stimulus. This code can be expected to be robust against possible subsequent noise sources, such as synaptic unreliability. The temporal pattern recognition task solved at the level of the ELL is particularly appropriate for computations which have to rely on the temporal resolution of up– and downstrokes, such as those underlying the jamming avoidance response.

## Acknowledgments

We thank Jenifer Juranek for computer assistance. Support: UCR and NSF grants, Center of Neuromorphic Systems Engineering as a part of the NSF ERC Program, and California Trade and Commerce Agency, Office of Strategic Technology.

## References

Anderson, T.W. (1984) An introduction to Multivariate Statistical Analysis. Wiley, New York.

Bastian, J. (1981) Electrolocation 2. The effects of moving objects and other electrical stimuli on the activities of two categories of posterior lateral line lobe cells in apteronotus albifrons. *J. Comp. Physiol. A*, **144**: 481-494.

Bialek, W., de Ruyter van Steveninck, R.R. & Warland, D. (1991) Reading a neural code. *Science*, **252**: 1854-1857.

Bullock, T.H. & Heiligengerg, W. (1986) Electroreception. Wiley, New York.

Carr, C.C., Maler, L. & Sas, E. (1982). Peripheral Organization and Central Projections of the Electrosensory Nerves in Gymnotiform Fish. *J. Comp. Neurol.*, **211**:139-153.

Gabbiani, F. & Koch, C. (1996) Coding of Time-Varying Signals in Spike Trains of Integrate-and-Fire Neurons with Random Threshold. *Neur. Comput.*, **8**: 44-66.

Gabbiani, F. (1996) Coding of time-varying signals in spike trains of linear and half-wave rectifying neurons. *Network: Comp. Neur. Syst.*, **7**:61-85.

Heiligenberg, W. (1991) Neural Nets in electric fish. MIT Press, Cambridge, MA.

Hopkins, C.D. (1976) Neuroethology of electric communication. *Ann. Rev. Neurosci.*, **11**:497-535.

Jolliffe, I.T. (1986) Principal Component Analysis. Springer-Verlag, New York.

Maler, L., Sas, E.K.B. & Rogers, J. (1981) The cytology of the posterior lateral line lobe of high-frequency weakly electric fish (gymnotidae): Dendritic differentiation and synaptic specificity. *J. Comp. Neurol.*, **255**: 526-537.

Metzner, W. (1993) The jamming avoidance response in Eigenmannia is controlled by two separate motor pathways. *J. Neurosci.*, **13**:1862-1878.

Metzner, W. & Heiligenberg, W. (1991). The coding of signals in the electric communication of the gymnotiform fish Eigenmannia: From electroreceptors to neurons in the torus semicircularis of the midbrain. *J. Comp. Physiol. A*, **169**: 135-150.

Poor, H.V. (1994) An introduction to Signal Detection and Estimation. Springer Verlag, New York.

Raudys, S.J. & Jain, A.K. (1991) Small sample size effects in statistical pattern recognition: Recommendations for practitioners. *IEEE Trans. Patt. Anal. Mach. Intell.*, **13**: 252-264.

Wessel, R., Koch, C. & Gabbiani F. (1996) Coding of Time-Varying Electric Field Amplitude Modulations in a Wave-Type Electric Fish *J. Neurophysiol.* **75**:2280-2293.

Zakon, H. (1986) The electroreceptive periphery. In: Bullock, T.H. & Heiligenberg, W. (eds), Electroreception, pp. 103-156. Wiley, New York.
